# A Large Deviation Bound
# for the Area Under the ROC Curve

**Shivani Agarwal**[*]**, Thore Graepel**[†]**, Ralf Herbrich**[†] **and Dan Roth**[*]

[*]Dept. of Computer Science
University of Illinois
Urbana, IL 61801, USA
{sagarwal,danr}@cs.uiuc.edu

[†]Microsoft Research
7 JJ Thomson Avenue
Cambridge CB3 0FB, UK
{thoreg,rherb}@microsoft.com

## Abstract

The area under the ROC curve (AUC) has been advocated as an evaluation criterion for the bipartite ranking problem. We study large deviation properties of the AUC; in particular, we derive a distribution-free large deviation bound for the AUC which serves to bound the expected accuracy of a ranking function in terms of its empirical AUC on an independent test sequence. A comparison of our result with a corresponding large deviation result for the classification error rate suggests that the test sample size required to obtain an $\epsilon$-accurate estimate of the expected accuracy of a ranking function with $\delta$-confidence is larger than that required to obtain an $\epsilon$-accurate estimate of the expected error rate of a classification function with the same confidence. A simple application of the union bound allows the large deviation bound to be extended to learned ranking functions chosen from finite function classes.

## 1   Introduction

In many learning problems, the goal is not simply to classify objects into one of a fixed number of classes; instead, a *ranking* of objects is desired. This is the case, for example, in information retrieval problems, where one is interested in retrieving documents from some database that are 'relevant' to a given query or topic. In such problems, one wants to return to the user a list of documents that contains relevant documents at the top and irrelevant documents at the bottom; in other words, one wants a ranking of the documents such that relevant documents are ranked higher than irrelevant documents.

The problem of ranking has been studied from a learning perspective under a variety of settings [2, 8, 4, 7]. Here we consider the setting in which objects come from two categories, positive and negative; the learner is given examples of objects labeled as positive or negative, and the goal is to learn a ranking in which positive objects are ranked higher than negative ones. This captures, for example, the information retrieval problem described above; in this case, the training examples consist of documents labeled as relevant (positive) or irrelevant (negative). This form of ranking problem corresponds to the 'bipartite feedback' case of [7]; for this reason, we refer to it as the *bipartite* ranking problem.

Formally, the setting of the bipartite ranking problem is similar to that of the binary classification problem. In both problems, there is an instance space $\mathcal{X}$ and a set of two class labels $\mathcal{Y} = \{-1, +1\}$. One is given a finite sequence of labeled training examples $S = ((\mathbf{x}_1, y_1), \ldots, (\mathbf{x}_M, y_M)) \in (\mathcal{X} \times \mathcal{Y})^M$, and the goal is to learn a function based on this training sequence. However, the form of the function to be learned in the two problems

is different. In classification, one seeks a binary-valued function $h : \mathcal{X} \rightarrow \mathcal{Y}$ that predicts the class of a new instance in $\mathcal{X}$. On the other hand, in ranking, one seeks a *real-valued* function $f : \mathcal{X} \rightarrow \mathbb{R}$ that induces a ranking over $\mathcal{X}$; an instance that is assigned a higher value by $f$ is ranked higher than one that is assigned a lower value by $f$.

The *area under the ROC curve* (AUC) has recently gained some attention as an evaluation criterion for the bipartite ranking [3]. Given a ranking function $f : \mathcal{X} \rightarrow \mathbb{R}$ and a finite data sequence $T = ((\mathbf{x}_1, y_1), \ldots, (\mathbf{x}_N, y_N)) \in (\mathcal{X} \times \mathcal{Y})^N$ containing $m$ positive and $n$ negative examples, the AUC of $f$ with respect to $T$, denoted $\hat{A}(f; T)$, can be expressed as the following Wilcoxon-Mann-Whitney statistic [3]:

$$\hat{A}(f; T) = \frac{1}{mn} \sum_{\{i: y_i = +1\}} \sum_{\{j: y_j = -1\}} \left( \mathbf{I}_{\{f(\mathbf{x}_i) > f(\mathbf{x}_j)\}} + \frac{1}{2} \mathbf{I}_{\{f(\mathbf{x}_i) = f(\mathbf{x}_j)\}} \right), \quad (1)$$

where $\mathbf{I}_{\{.\}}$ denotes the indicator variable whose value is one if its argument is true and zero otherwise. The AUC of $f$ with respect to $T$ is thus simply the fraction of positive-negative pairs in $T$ that are ranked correctly by $f$, assuming that ties are broken uniformly at random.[1]

The AUC is an empirical quantity that evaluates a ranking function with respect to a particular data sequence. What does the empirical AUC tell us about the expected performance of a ranking function on future examples? This is the question we consider. The question has two parts, both of which are important for machine learning practice. First, what can be said about the expected performance of a ranking function based on its empirical AUC on an independent test sequence? Second, what can be said about the expected performance of a learned ranking function based on its empirical AUC on the training sequence from which it is learned? We address the first question in this paper; the second question is addressed in [1].

We start by defining the expected ranking accuracy of a ranking function (analogous to the expected error rate of a classification function) in Section 2. Section 3 contains our large deviation result, which serves to bound the expected accuracy of a ranking function in terms of its empirical AUC on an independent test sequence. Our conceptual approach in deriving the large deviation result for the AUC is similar to that of [9], in which large deviation properties of the average precision were considered. Section 4 compares our bound to a corresponding large deviation bound for the classification error rate. A simple application of the union bound allows the large deviation bound to be extended to learned ranking functions chosen from finite function classes; this is described in Section 5.

## 2   Expected Ranking Accuracy

We begin by introducing some notation. As in classification, we shall assume that all examples are drawn randomly and independently according to some (unknown) underlying distribution $\mathcal{D}$ over $\mathcal{X} \times \mathcal{Y}$. The notation $\mathcal{D}_{+1}$ and $\mathcal{D}_{-1}$ will be used to denote the class-conditional distributions $\mathcal{D}_{X|Y=+1}$ and $\mathcal{D}_{X|Y=-1}$, respectively. We shall find it convenient to decompose a data sequence $T = ((\mathbf{x}_1, y_1), \ldots, (\mathbf{x}_N, y_N)) \in (\mathcal{X} \times \mathcal{Y})^N$ into two components, $T_X = (\mathbf{x}_1, \ldots, \mathbf{x}_N) \in \mathcal{X}^N$ and $T_Y = (y_1, \ldots, y_N) \in \mathcal{Y}^N$. Several of our results will involve the conditional distribution $\mathcal{D}_{T_X|T_Y=\underline{y}}$ for some label sequence $\underline{y} = (y_1, \ldots, y_N) \in \mathcal{Y}^N$; this distribution is simply $\mathcal{D}_{y_1} \times \ldots \times \mathcal{D}_{y_N}$.[2] As a final note of

convention, we use $T \in (\mathcal{X} \times \mathcal{Y})^N$ to denote a general data sequence (*e.g.*, an independent test sequence), and $S \in (\mathcal{X} \times \mathcal{Y})^M$ to denote a training sequence.

**Definition 1 (Expected ranking accuracy).** *Let $f : \mathcal{X} \rightarrow \mathbb{R}$ be a ranking function on $\mathcal{X}$. Define the* expected ranking accuracy *(or simply* ranking accuracy*) of $f$, denoted by $A(f)$, as follows:*

$$A(f) \quad = \quad \mathbf{E}_{X \sim \mathcal{D}_{+1}, X' \sim \mathcal{D}_{-1}} \left\{ \mathbf{I}_{\{f(X) > f(X')\}} + \frac{1}{2} \mathbf{I}_{\{f(X) = f(X')\}} \right\}.$$

The ranking accuracy $A(f)$ defined above is simply the probability that an instance drawn randomly according to $\mathcal{D}_{+1}$ will be ranked higher by $f$ than an instance drawn randomly according to $\mathcal{D}_{-1}$, assuming that ties are broken uniformly at random. The following simple lemma shows that the empirical AUC of a ranking function $f$ is an unbiased estimator of the expected ranking accuracy of $f$:

**Lemma 1.** *Let $f : \mathcal{X} \rightarrow \mathbb{R}$ be a ranking function on $\mathcal{X}$, and let $\underline{y} = (y_1, \ldots, y_N) \in \mathcal{Y}^N$ be any finite label sequence. Then*

$$\mathbf{E}_{T_X | T_Y = \underline{y}} \left\{ \hat{A}(f; T) \right\} \quad = \quad A(f).$$

*Proof.* Let $m$ be the number of positive labels in $\underline{y}$, and $n$ the number of negative labels in $\underline{y}$. Then from the definition of the AUC (Eq. (1)) and linearity of expectation, we have

$$
\begin{aligned}
\mathbf{E}_{T_X | T_Y = \underline{y}} &\left\{ \hat{A}(f; T) \right\} \\
&= \quad \frac{1}{mn} \sum_{\{i:y_i=+1\}} \sum_{\{j:y_j=-1\}} \mathbf{E}_{X_i \sim \mathcal{D}_{+1}, X_j \sim \mathcal{D}_{-1}} \left\{ \mathbf{I}_{\{f(X_i) > f(X_j)\}} + \frac{1}{2} \mathbf{I}_{\{f(X_i) = f(X_j)\}} \right\} \\
&= \quad \frac{1}{mn} \sum_{\{i:y_i=+1\}} \sum_{\{j:y_j=-1\}} A(f) \\
&= \quad A(f).
\end{aligned}
$$
$\square$

## 3    Large Deviation Bound

We are interested in bounding the probability that the empirical AUC of a ranking function $f$ with respect to a (random) test sequence $T$ will have a large deviation from its expected ranking accuracy. In other words, we are interested in bounding probabilities of the form

$$\mathbf{P} \left\{ \left| \hat{A}(f; T) - A(f) \right| \geq \epsilon \right\}$$

for given $\epsilon > 0$. Our main tool in deriving such a large deviation bound will be the following powerful concentration inequality of McDiarmid [10], which bounds the deviation of any function of a sample for which a single change in the sample has limited effect:

**Theorem 1 (McDiarmid, 1989).** *Let $X_1, \ldots, X_N$ be independent random variables with $X_k$ taking values in a set $A_k$ for each $k$. Let $\phi : (A_1 \times \cdots \times A_N) \rightarrow \mathbb{R}$ be such that*

$$\sup_{x_i \in A_i, x'_k \in A_k} |\phi(x_1, \ldots, x_N) - \phi(x_1, \ldots, x_{k-1}, x'_k, x_{k+1}, \ldots, x_N)| \quad \leq \quad c_k.$$

*Then for any $\epsilon > 0$,*

$$\mathbf{P} \left\{ |\phi(X_1, \ldots, X_N) - \mathbf{E}\{\phi(X_1, \ldots, X_N)\}| \geq \epsilon \right\} \quad \leq \quad 2e^{-2\epsilon^2 / \sum_{k=1}^N c_k^2}.$$

Note that when $X_1, \ldots, X_N$ are independent bounded random variables with $X_k \in [a_k, b_k]$ with probability one and $\phi(X_1, \ldots, X_N) = \sum_{k=1}^N X_k$, McDiarmid's inequality (with $c_k = b_k - a_k$) reduces to Hoeffding's inequality. Next we define the following quantity which appears in several of our results:

**Definition 2 (Positive skew).** *Let* $\underline{y} = (y_1, \ldots, y_N) \in \mathcal{Y}^N$ *be a finite label sequence of length* $N \in \mathbb{N}$. *Define the* positive skew *of* $\underline{y}$, *denoted by* $\rho(\underline{y})$, *as follows:*

$$\rho(\underline{y}) \;=\; \frac{1}{N} \sum_{\{i : y_i = +1\}} 1 \,.$$

The following can be viewed as the main result of this paper. We note that our results are all distribution-free, in the sense that they hold for any distribution $\mathcal{D}$ over $\mathcal{X} \times \mathcal{Y}$.

**Theorem 2.** *Let* $f : \mathcal{X} \to \mathbb{R}$ *be a fixed ranking function on* $\mathcal{X}$ *and let* $\underline{y} = (y_1, \ldots, y_N) \in \mathcal{Y}^N$ *be any label sequence of length* $N \in \mathbb{N}$. *Then for any* $\epsilon > 0$,

$$\mathbf{P}_{T_X | T_Y = \underline{y}} \left\{ \left| \hat{A}(f; T) - A(f) \right| \geq \epsilon \right\} \;\leq\; 2 e^{-2\rho(\underline{y})(1 - \rho(\underline{y}))N\epsilon^2} \,.$$

*Proof.* Let $m$ be the number of positive labels in $\underline{y}$, and $n$ the number of negative labels in $\underline{y}$. We can view $T_X = (X_1, \ldots, X_N) \in \mathcal{X}^N$ as a random vector; given the label sequence $\underline{y}$, the random variables $X_1, \ldots, X_N$ are independent, with each $X_k$ taking values in $\mathcal{X}$. Now, define $\phi : \mathcal{X}^N \to \mathbb{R}$ as follows:

$$\phi(\mathbf{x}_1, \ldots, \mathbf{x}_N) \;=\; \hat{A}\left( f; ((\mathbf{x}_1, y_1), \ldots, (\mathbf{x}_N, y_N)) \right) \,.$$

Then, for each $k$ such that $y_k = +1$, we have the following for all $\mathbf{x}_i, \mathbf{x}'_k \in \mathcal{X}$:

$$\left| \phi(\mathbf{x}_1, \ldots, \mathbf{x}_N) - \phi(\mathbf{x}_1, \ldots, \mathbf{x}_{k-1}, \mathbf{x}'_k, \mathbf{x}_{k+1} \ldots, \mathbf{x}_N) \right|$$

$$= \frac{1}{mn} \left| \sum_{\{j : y_j = -1\}} \left( \left( \mathbf{I}_{\{f(\mathbf{x}_k) > f(\mathbf{x}_j)\}} + \frac{1}{2}\mathbf{I}_{\{f(\mathbf{x}_k) = f(\mathbf{x}_j)\}} \right) - \right. \right.$$

$$\left. \left. \left( \mathbf{I}_{\{f(\mathbf{x}'_k) > f(\mathbf{x}_j)\}} + \frac{1}{2}\mathbf{I}_{\{f(\mathbf{x}'_k) = f(\mathbf{x}_j)\}} \right) \right) \right|$$

$$\leq \frac{1}{mn} n$$

$$= \frac{1}{m} \,.$$

Similarly, for each $k$ such that $y_k = -1$, one can show for all $\mathbf{x}_i, \mathbf{x}'_k \in \mathcal{X}$:

$$\left| \phi(\mathbf{x}_1, \ldots, \mathbf{x}_N) - \phi(\mathbf{x}_1, \ldots, \mathbf{x}_{k-1}, \mathbf{x}'_k, \mathbf{x}_{k+1} \ldots, \mathbf{x}_N) \right| \;\leq\; \frac{1}{n} \,.$$

Thus, taking $c_k = 1/m$ for $k$ such that $y_k = +1$ and $c_k = 1/n$ for $k$ such that $y_k = -1$, and applying McDiarmid's theorem, we get for any $\epsilon > 0$,

$$\mathbf{P}_{T_X | T_Y = \underline{y}} \left\{ \left| \hat{A}(f; T) - \mathbf{E}_{T_X | T_Y = \underline{y}} \left\{ \hat{A}(f; T) \right\} \right| \geq \epsilon \right\} \;\leq\; 2 e^{-2\epsilon^2 / (m(\frac{1}{m})^2 + n(\frac{1}{n})^2)} \,. \quad (2)$$

Now, from Lemma 1,

$$\mathbf{E}_{T_X | T_Y = \underline{y}} \left\{ \hat{A}(f; T) \right\} \;=\; A(f) \,.$$

Also, we have

$$\frac{1}{m(\frac{1}{m})^2 + n(\frac{1}{n})^2} \;=\; \frac{1}{\frac{1}{m} + \frac{1}{n}} \;=\; \frac{mn}{m+n} \;=\; \rho(\underline{y})(1 - \rho(\underline{y}))N \,.$$

Substituting the above in Eq. (2) gives the desired result. $\qquad\square$

We note that the result of Theorem 2 can be strengthened so that the conditioning is only on the numbers $m$ and $n$ of positive and negative labels, and not on the specific label vector $\underline{y}$.[3] From Theorem 2, we can derive a confidence interval interpretation of the bound that gives, for any $0 < \delta \leq 1$, a confidence interval based on the empirical AUC of a ranking function (on a random test sequence) which is likely to contain the true ranking accuracy with probability at least $1 - \delta$. More specifically, we have:

**Corollary 1.** *Let $f : \mathcal{X} \to \mathbb{R}$ be a fixed ranking function on $\mathcal{X}$ and let $\underline{y} = (y_1, \ldots, y_N) \in \mathcal{Y}^N$ be any label sequence of length $N \in \mathbb{N}$. Then for any $0 < \delta \leq 1$,*

$$\mathbf{P}_{T_X | T_Y = \underline{y}} \left\{ \left| \hat{A}(f; T) - A(f) \right| \geq \sqrt{\frac{\ln\left(\frac{2}{\delta}\right)}{2\rho(\underline{y})(1 - \rho(\underline{y}))N}} \right\} \leq \delta.$$

*Proof.* This follows directly from Theorem 2 by setting $2e^{-2\rho(\underline{y})(1-\rho(\underline{y}))N\epsilon^2} = \delta$ and solving for $\epsilon$. □

Theorem 2 also allows us to obtain an expression for a test sample size that is sufficient to obtain, for $0 < \epsilon, \delta \leq 1$, an $\epsilon$-accurate estimate of the ranking accuracy with $\delta$-confidence:

**Corollary 2.** *Let $f : \mathcal{X} \to \mathbb{R}$ be a fixed ranking function on $\mathcal{X}$ and let $0 < \epsilon, \delta \leq 1$. Let $\underline{y} = (y_1, \ldots, y_N) \in \mathcal{Y}^N$ be any label sequence of length $N \in \mathbb{N}$. If*

$$N \geq \frac{\ln\left(\frac{2}{\delta}\right)}{2\rho(\underline{y})(1 - \rho(\underline{y}))\epsilon^2},$$

*then*

$$\mathbf{P}_{T_X | T_Y = \underline{y}} \left\{ \left| \hat{A}(f; T) - A(f) \right| \geq \epsilon \right\} \leq \delta.$$

*Proof.* This follows directly from Theorem 2 by setting $2e^{-2\rho(\underline{y})(1-\rho(\underline{y}))N\epsilon^2} \leq \delta$ and solving for $N$. □

Figure 1 illustrates the dependence of the above expression for the sufficient test sample size on the the accuracy parameter $\epsilon$ and positive skew $\rho(\underline{y})$ for different values of $\delta$.

The confidence interval of Corollary 1 can in fact be generalized to remove the conditioning on the label vector completely:

**Theorem 3.** *Let $f : \mathcal{X} \to \mathbb{R}$ be a fixed ranking function on $\mathcal{X}$ and let $N \in \mathbb{N}$. Then for any $0 < \delta \leq 1$,*

$$\mathbf{P}_{T \sim \mathcal{D}^N} \left\{ \left| \hat{A}(f; T) - A(f) \right| \geq \sqrt{\frac{\ln\left(\frac{2}{\delta}\right)}{2\rho(T_Y)(1 - \rho(T_Y))N}} \right\} \leq \delta.$$

*Proof.* For $T \in (\mathcal{X} \times \mathcal{Y})^N$ and $0 < \delta \leq 1$, define the proposition

$$\Phi(T, \delta) \equiv \left\{ \left| \hat{A}(f; T) - A(f) \right| \geq \sqrt{\frac{\ln\left(\frac{2}{\delta}\right)}{2\rho(T_Y)(1 - \rho(T_Y))N}} \right\}.$$

Then for any $0 < \delta \leq 1$, we have

$$
\begin{aligned}
\mathbf{P}_T \left\{ \Phi(T, \delta) \right\} &= \mathbf{E}_T \left\{ \mathbf{I}_{\Phi(T,\delta)} \right\} \\
&= \mathbf{E}_{T_Y} \left\{ \mathbf{E}_{T_X | T_Y = \underline{y}} \left\{ \mathbf{I}_{\Phi(T,\delta)} \right\} \right\} \\
&= \mathbf{E}_{T_Y} \left\{ \mathbf{P}_{T_X | T_Y = \underline{y}} \left\{ \Phi(T, \delta) \right\} \right\} \\
&\leq \mathbf{E}_{T_Y} \left\{ \delta \right\} \quad \text{(by Corollary 1)} \\
&= \delta.
\end{aligned}
$$
□

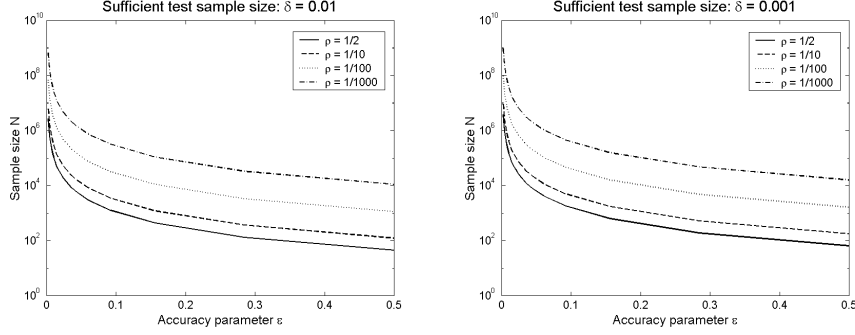

Figure 1: The test sample size $N$ (based on Corollary 2) sufficient to obtain an $\epsilon$-accurate estimate of the ranking accuracy with $\delta$-confidence, for various values of the positive skew $\rho \equiv \rho(\underline{y})$ for some label sequence $\underline{y}$, for (left) $\delta = 0.01$ and (right) $\delta = 0.001$.

Note that the above 'trick' works only once we have gone to a confidence interval; an attempt to generalize the bound of Theorem 2 in a similar way gives an expression in which the final expectation is not easy to evaluate. Interestingly, the above proof does not even require a factorized distribution $\mathcal{D}_{T_Y}$ since it is built on a result for any fixed label sequence $\underline{y}$. We note that the above technique could also be applied to generalize the results of [9] in a similar manner.

## 4   Comparison with Large Deviation Bound for Error Rate

Our use of McDiarmid's inequality in deriving the large deviation bound for the AUC of a ranking function is analogous to the use of Hoeffding's inequality in deriving a large deviation bound for the error rate of a classification function. (*e.g.*, see [6, Chapter 8]). The need for the more general inequality of McDiarmid in our derivations arises from the fact that the empirical AUC, unlike the empirical error rate, cannot be expressed as a sum of independent random variables.

Given a classification function $h : \mathcal{X} \rightarrow \mathcal{Y}$, let $L(h)$ denote the expected error rate of $h$:

$$L(h) \;\; = \;\; \mathbf{E}_{XY \sim \mathcal{D}} \left\{ \mathbf{I}_{\{h(X) \neq Y\}} \right\} .$$

Similarly, given a classification function $h : \mathcal{X} \rightarrow \mathcal{Y}$ and a finite data sequence $T = ((\mathbf{x}_1, y_1), \ldots, (\mathbf{x}_N, y_N)) \in (\mathcal{X} \times \mathcal{Y})^N$, let $\hat{L}(h; T)$ denote the empirical error rate of $h$ with respect to $T$:

$$\hat{L}(h; T) \;\; = \;\; \frac{1}{N} \sum_{i=1}^{N} \mathbf{I}_{\{h(\mathbf{x}_i) \neq y_i\}} .$$

Then the large deviation bound obtained via Hoeffding's inequality for the classification error rate states that for a fixed classification function $h : \mathcal{X} \rightarrow \mathcal{Y}$ and for any $N \in \mathbb{N}$, $\epsilon > 0$,

$$\mathbf{P}_{T \sim \mathcal{D}^N} \left\{ \left| \hat{L}(h; T) - L(h) \right| \geq \epsilon \right\} \;\; \leq \;\; 2 e^{-2N\epsilon^2} . \tag{3}$$

Comparing Eq. (3) to the bound of Theorem 2, we see that the AUC bound differs from the error rate bound by a factor of $\rho(\underline{y})(1 - \rho(\underline{y}))$ in the exponent. This difference translates into a $1/(\rho(\underline{y})(1 - \rho(\underline{y})))$ factor difference in the resulting sample size bounds: given $0 < \epsilon, \delta \leq 1$, the test sample size sufficient to obtain an $\epsilon$-accurate estimate of the expected accuracy of a ranking function with $\delta$-confidence is $1/(\rho(\underline{y})(1-\rho(\underline{y})))$ times larger than the corresponding test sample size sufficient to obtain an $\epsilon$-accurate estimate of the expected error rate of a classification function with the same confidence. For $\rho(\underline{y}) = 1/2$, this means a sample size larger by a factor of 4; as the positive skew $\rho(\underline{y})$ departs from $1/2$, the factor grows larger (see Figure 2).

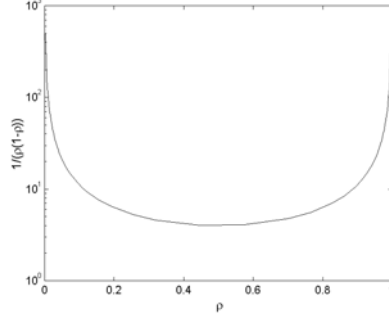

Figure 2: The test sample size bound for the AUC, for positive skew $\rho \equiv \rho(\underline{y})$ for some label sequence $\underline{y}$, is larger than the corresponding test sample size bound for the classification error rate by a factor of $1/(\rho(1-\rho))$.

## 5   Bound for Learned Ranking Functions Chosen from Finite Classes

The large deviation result of Theorem 2 bounds the expected accuracy of a ranking function in terms of its empirical AUC on an independent test sequence. A simple application of the union bound allows the result to be extended to bound the expected accuracy of a learned ranking function in terms of its empirical AUC on the training sequence from which it is learned, in the case when the learned ranking function is chosen from a finite function class. More specifically, we have:

**Theorem 4.** *Let $\mathcal{F}$ be a finite class of real-valued functions on $\mathcal{X}$ and let $f_S \in \mathcal{F}$ denote the ranking function chosen by a learning algorithm based on the training sequence $S$. Let $\underline{y} = (y_1, \ldots, y_M) \in \mathcal{Y}^M$ be any label sequence of length $M \in \mathbb{N}$. Then for any $\epsilon > 0$,*

$$\mathbf{P}_{S_X|S_Y=\underline{y}}\left\{\left|\hat{A}(f_S; S) - A(f_S)\right| \geq \epsilon\right\} \;\; \leq \;\; 2|\mathcal{F}|e^{-2\rho(\underline{y})(1-\rho(\underline{y}))M\epsilon^2}\,.$$

*Proof.*  For any $\epsilon > 0$, we have

$$\mathbf{P}_{S_X|S_Y=\underline{y}}\left\{\left|\hat{A}(f_S; S) - A(f_S)\right| \geq \epsilon\right\}$$

$$\leq \;\; \mathbf{P}_{S_X|S_Y=\underline{y}}\left\{\max_{f\in\mathcal{F}}\left|\hat{A}(f; S) - A(f)\right| \geq \epsilon\right\}$$

$$\leq \;\; \sum_{f\in\mathcal{F}}\mathbf{P}_{S_X|S_Y=\underline{y}}\left\{\left|\hat{A}(f; S) - A(f)\right| \geq \epsilon\right\} \qquad \text{(by the union bound)}$$

$$\leq \;\; 2|\mathcal{F}|e^{-2\rho(\underline{y})(1-\rho(\underline{y}))M\epsilon^2} \qquad \text{(by Theorem 2)}\,. \qquad\qquad \square$$

As before, we can derive from Theorem 4 expressions for confidence intervals and sufficient training sample size. We give these here without proof:

**Corollary 3.** *Under the assumptions of Theorem 4, for any $0 < \delta \leq 1$,*

$$\mathbf{P}_{S_X|S_Y=\underline{y}}\left\{\left|\hat{A}(f_S; S) - A(f_S)\right| \geq \sqrt{\frac{\ln|\mathcal{F}| + \ln\left(\frac{2}{\delta}\right)}{2\rho(\underline{y})(1-\rho(\underline{y}))M}}\right\} \;\; \leq \;\; \delta\,.$$

**Corollary 4.** *Under the assumptions of Theorem 4, for any $0 < \epsilon, \delta \leq 1$, if*

$$M \;\; \geq \;\; \frac{1}{2\rho(\underline{y})(1-\rho(\underline{y}))\epsilon^2}\left(\ln|\mathcal{F}| + \ln\left(\frac{2}{\delta}\right)\right),$$

*then*

$$\mathbf{P}_{S_X|S_Y=\underline{y}}\left\{\left|\hat{A}(f_S;S)-A(f_S)\right|\geq\epsilon\right\} \quad\leq\quad \delta\,.$$

**Theorem 5.** *Let $\mathcal{F}$ be a finite class of real-valued functions on $\mathcal{X}$ and let $f_S \in \mathcal{F}$ denote the ranking function chosen by a learning algorithm based on the training sequence $S$. Let $M \in \mathbb{N}$. Then for any $0 < \delta \leq 1$,*

$$\mathbf{P}_{S\sim\mathcal{D}^M}\left\{\left|\hat{A}(f_S;S)-A(f_S)\right|\geq\sqrt{\frac{\ln|\mathcal{F}|+\ln\left(\frac{2}{\delta}\right)}{2\rho(S_Y)(1-\rho(S_Y))M}}\right\} \quad\leq\quad \delta\,.$$

## 6   Conclusion

We have derived a distribution-free large deviation bound for the area under the ROC curve (AUC), a quantity used as an evaluation criterion for the bipartite ranking problem. Our result parallels the classical large deviation result for the classification error rate obtained via Hoeffding's inequality. Since the AUC cannot be expressed as a sum of independent random variables, a more powerful inequality of McDiarmid was required. A comparison with the corresponding large deviation result for the error rate suggests that, in the distribution-free setting, the test sample size required to obtain an $\epsilon$-accurate estimate of the expected accuracy of a ranking function with $\delta$-confidence is larger than the test sample size required to obtain a similar estimate of the expected error rate of a classification function. A simple application of the union bound allows the large deviation bound to be extended to learned ranking functions chosen from finite function classes.

A possible route for deriving an alternative large deviation bound for the AUC could be via the theory of U-statistics; the AUC can be expressed as a two-sample U-statistic, and therefore it may be possible to apply specialized results from U-statistic theory (see, for example, [5]) to the AUC.

## Footnotes

[1]In [3], a slightly simpler form of the Wilcoxon-Mann-Whitney statistic is used, which does not account for ties.

[2]Note that, since the AUC of a ranking function $f$ with respect to a data sequence $T \in (\mathcal{X} \times \mathcal{Y})^N$ is independent of the ordering of examples in the sequence, our results involving the conditional distribution $\mathcal{D}_{T_X|T_Y=\underline{y}}$ for some label sequence $\underline{y} = (y_1, \ldots, y_N) \in \mathcal{Y}^N$ depend only on the number $m$ of positive labels in $\underline{y}$ and the number $n$ of negative labels in $\underline{y}$. We state our results in terms of the distribution $\mathcal{D}_{T_X|T_Y=\underline{y}} \equiv \mathcal{D}_{y_1} \times \ldots \times \mathcal{D}_{y_N}$ only because this is more general than $\mathcal{D}_{+1}^m \times \mathcal{D}_{-1}^n$.

[3]Our thanks to an anonymous reviewer for pointing this out.

## References

[1] S. Agarwal, S. Har-Peled, and D. Roth. A uniform convergence bound for the area under the ROC curve. In *Proceedings of the 10th International Workshop on Artificial Intelligence and Statistics*, 2005.

[2] W. W. Cohen, R. E. Schapire, and Y. Singer. Learning to order things. *Journal of Artificial Intelligence Research*, 10:243–270, 1999.

[3] C. Cortes and M. Mohri. AUC optimization vs. error rate minimization. In S. Thrun, L. Saul, and B. Schölkopf, editors, *Advances in Neural Information Processing Systems 16*, 2004.

[4] K. Crammer and Y. Singer. Pranking with ranking. In T. G. Dietterich, S. Becker, and Z. Ghahramani, editors, *Advances in Neural Information Processing Systems 14*, 2002.

[5] V. H. de la Pẽna and E. Giné. *Decoupling: From Dependence to Independence*. Springer-Verlag, New York, 1999.

[6] L. Devroye, L. Györfi, and G. Lugosi. *A Probabilistic Theory of Pattern Recognition*. Springer-Verlag, New York, 1996.

[7] Y. Freund, R. Iyer, R. E. Schapire, and Y. Singer. An efficient boosting algorithm for combining preferences. *Journal of Machine Learning Research*, 4:933–969, 2003.

[8] R. Herbrich, T. Graepel, and K. Obermayer. Large margin rank boundaries for ordinal regression. *Advances in Large Margin Classifiers*, pages 115–132, 2000.

[9] S. I. Hill, H. Zaragoza, R. Herbrich, and P. J. W. Rayner. Average precision and the problem of generalisation. In *Proceedings of the ACM SIGIR Workshop on Mathematical and Formal Methods in Information Retrieval*, 2002.

[10] C. McDiarmid. On the method of bounded differences. In *Surveys in Combinatorics 1989*, pages 148–188. Cambridge University Press, 1989.
